# Real-Time Pitch Determination of One or More Voices by Nonnegative Matrix Factorization

**Fei Sha and Lawrence K. Saul**
Dept. of Computer and Information Science
University of Pennsylvania, Philadelphia, PA 19104
{*feisha,lsaul*}*@cis.upenn.edu*

## Abstract

An auditory "scene", composed of overlapping acoustic sources, can be viewed as a complex object whose constituent parts are the individual sources. Pitch is known to be an important cue for auditory scene analysis. In this paper, with the goal of building agents that operate in human environments, we describe a real-time system to identify the presence of one or more voices and compute their pitch. The signal processing in the front end is based on instantaneous frequency estimation, a method for tracking the partials of voiced speech, while the pattern-matching in the back end is based on nonnegative matrix factorization, an unsupervised algorithm for learning the parts of complex objects. While supporting a framework to analyze complicated auditory scenes, our system maintains real-time operability and state-of-the-art performance in clean speech.

## 1 Introduction

Nonnegative matrix factorization (NMF) is an unsupervised algorithm for learning the parts of complex objects [11]. The algorithm represents high dimensional inputs ("objects") by a linear superposition of basis functions ("parts") in which both the linear coefficients and basis functions are constrained to be nonnegative. Applied to images of faces, NMF learns basis functions that correspond to eyes, noses, and mouths; applied to handwritten digits, it learns basis functions that correspond to cursive strokes. The algorithm has also been implemented in real-time embedded systems as part of a visual front end [10].

Recently, it has been suggested that NMF can play a similarly useful role in speech and audio processing [16, 17]. An auditory "scene", composed of overlapping acoustic sources, can be viewed as a complex object whose constituent parts are the individual sources. Pitch is known to be an extremely important cue for source separation and auditory scene analysis [4]. It is also an acoustic cue that seems amenable to modeling by NMF. In particular, we can imagine the basis functions in NMF as harmonic stacks of individual periodic sources (e.g., voices, instruments), which are superposed to give the magnitude spectrum of a mixed signal. The pattern-matching computations of NMF are reminiscent of long-standing template-based models of pitch perception [6].

Our interest in NMF lies mainly in its use for speech processing. In this paper, we describe a real-time system to detect the presence of one or more voices and determine their pitch.

Learning plays a crucial role in our system: the basis functions of NMF are trained offline from data to model the particular timbres of voiced speech, which vary across different phonetic contexts and speakers. In related work, Smaragdis and Brown used NMF to model polyphonic piano music [17]. Our work differs in its focus on speech, real-time processing, and statistical learning of basis functions.

A long-term goal is to develop interactive voice-driven agents that respond to the pitch contours of human speech [15]. To be truly interactive, these agents must be able to process input from distant sources and to operate in noisy environments with overlapping speakers. In this paper, we have taken an important step toward this goal by maintaining real-time operability and state-of-the-art performance in clean speech while developing a framework that can analyze more complicated auditory scenes. These are inherently *competing* goals in engineering. Our focus on actual system-building also distinguishes our work from many other studies of overlapping periodic sources [5, 9, 19, 20, 21].

The organization of this paper is as follows. In section 2, we describe the signal processing in our front end that converts speech signals into a form that can be analyzed by NMF. In section 3, we describe the use of NMF for pitch tracking—namely, the learning of basis functions for voiced speech, and the nonnegative deconvolution for real-time analysis. In section 4, we present experimental results on signals with one or more voices. Finally, in section 5, we conclude with plans for future work.

## 2    Signal processing

A periodic signal is characterized by its fundamental frequency, $f_0$. It can be decomposed by Fourier analysis as the sum of sinusoids—or partials—whose frequencies occur at integer multiples of $f_0$. For periodic signals with unknown $f_0$, the frequencies of the partials can be inferred from peaks in the magnitude spectrum, as computed by an FFT.

Voiced speech is perceived as having a pitch at the fundamental frequency of vocal cord vibration. Perfect periodicity is an idealization, however; the waveforms of voiced speech are non-stationary, quasiperiodic signals. In practice, one cannot reliably extract the partials of voiced speech by simply computing windowed FFTs and locating peaks in the magnitude spectrum. In this section, we review a more robust method, known as instantaneous frequency (IF) estimation [1], for extracting the stable sinusoidal components of voiced speech. This method is the basis for the signal processing in our front-end.

The starting point of IF estimation is to model the voiced speech signal, $s(t)$, by a sum of amplitude and frequency-modulated sinusoids:

$$s(t) = \sum_i \alpha_i(t) \cos \left( \int_0^t dt\, \omega_i(t) + \theta_i \right). \tag{1}$$

The arguments of the cosines in eq. (1) are called the instantaneous phases; their derivatives with respect to time yield the so-called instantaneous frequencies $\omega_i(t)$. If the amplitudes $\alpha_i(t)$ and frequencies $\omega_i(t)$ are stationary, then eq. (1) reduces to a weighted sum of pure sinusoids. For nonstationary signals, $\omega_i(t)$ intuitively represents the instantaneous frequency of the $i$th partial at time $t$.

The short-time Fourier transform (STFT) provides an efficient tool for IF estimation [2]. The STFT of $s(t)$ with windowing function $w(t)$ is given by:

$$F(\omega, t) = \int d\tau\, s(\tau) w(\tau - t) e^{-j\omega\tau}. \tag{2}$$

Let $z(\omega, t) = e^{j\omega t} F(\omega, t)$ denote the analytic signal of the Fourier component of $s(t)$ with frequency $\omega$, and let $a = \mathrm{Re}[z]$ and $b = \mathrm{Im}[z]$ denote its real and imaginary parts. We

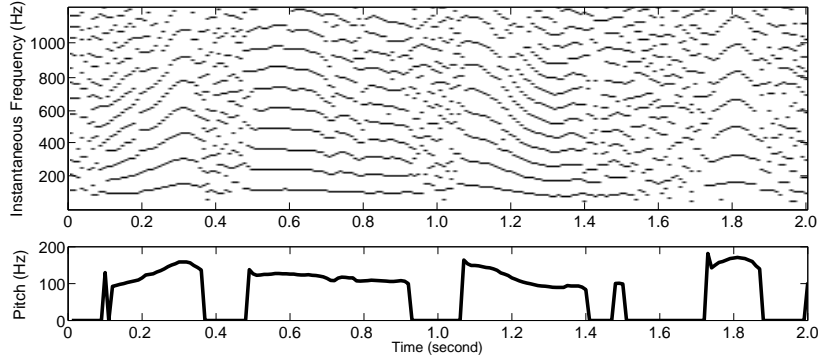

Figure 1: Top: instantaneous frequencies of estimated partials for the utterance "The north wind and the sun were disputing." Bottom: $f_0$ contour derived from a laryngograph recording.

can define a mapping from the time-frequency plane of the STFT to another frequency axis $\lambda(\omega, t)$ by:

$$\lambda(\omega, t) = \frac{\partial}{\partial t} \arg[z(\omega, t)] = \frac{a\frac{\partial b}{\partial t} - b\frac{\partial a}{\partial t}}{a^2 + b^2} \tag{3}$$

The derivatives on the right hand side can be computed efficiently via SFFTs [2]. Note that the right hand side of eq. (3) differentiates the instantaneous phase associated with a particular Fourier component of $s(t)$. IF estimation identifies the stable fixed points [7, 8] of this mapping, given by

$$\lambda(\omega^*, t) = \omega^* \quad \text{and} \quad (\partial\lambda/\partial\omega)|_{\omega=\omega^*} < 1, \tag{4}$$

as the instantaneous frequencies of the partials that appear in eq. (1). Intuitively, these fixed points occur where the notions of energy at frequency $\omega$ in eqs. (1) and (2) coincide—that is, where the IF and STFT representations appear most consistent.

The top panel of Fig. 1 shows the IFs of partials extracted by this method for a speech signal with sliding and overlapping analysis windows. The bottom panels shows the pitch contour. Note that in regions of voiced speech, indicated by nonzero $f_0$ values, the IFs exhibit a clear harmonic structure, while in regions of unvoiced speech, they do not.

In summary, the signal processing in our front-end extracts partials with frequencies $\omega_i^*(t)$ and nonnegative amplitudes $|F(\omega_i^*(t), t)|$, where $t$ indexes the time of the analysis window and $i$ indexes the number of extracted partials. Further analysis of the signal is performed by the NMF algorithm described in the next section, which is used to detect the presence of one or more voices and to estimate their $f_0$ values. Similar front ends have been used in other studies of pitch tracking and source separation [1, 2, 7, 13].

## 3 Nonnegative matrix factorization

For mixed signals of overlapping speakers, our front-end outputs the mixture of partials extracted from several voices. How can we analyze this output by NMF? In this section, we show: (i) how to learn nonnegative basis functions that model the characteristic timbres of voiced speech, and (ii) how to decompose mixed signals in terms of these basis functions.

We briefly review NMF [11]. Given observations $\mathbf{y}_t$, the goal of NMF is to compute basis functions $\mathbf{W}$ and linear coefficients $\mathbf{x}_t$ such that the reconstructed vectors $\tilde{\mathbf{y}}_t = \mathbf{W}\mathbf{x}_t$

best match the original observations. The observations, basis functions, and coefficients are constrained to be nonnegative. Reconstruction errors are measured by the generalized Kullback-Leibler divergence:

$$G(\mathbf{y}, \tilde{\mathbf{y}}) = \sum_{\alpha} \left[ y_\alpha \log(y_\alpha/\tilde{y}_\alpha) - y_\alpha + \tilde{y}_\alpha \right], \tag{5}$$

which is lower bounded by zero and vanishes if and only if $\mathbf{y} = \tilde{\mathbf{y}}$. NMF works by optimizing the total reconstruction error $\sum_t G(\mathbf{y}_t, \tilde{\mathbf{y}}_t)$ in terms of the basis functions $\mathbf{W}$ and coefficients $\mathbf{x}_t$. We form three matrices by concatenating the column vectors $\mathbf{y}_t$, $\tilde{\mathbf{y}}_t$ and $\mathbf{x}_t$ separately and denote them by $\mathbf{Y}$, $\tilde{\mathbf{Y}}$ and $\mathbf{X}$ respectively. Multiplicative updates for the optimization problem are given in terms of the elements of these matrices:

$$W_{\alpha\beta} \leftarrow W_{\alpha\beta} \left[ \sum_t X_{\beta t} \left( \frac{Y_{\alpha t}}{\tilde{Y}_{\alpha t}} \right) \right], \qquad X_{\beta t} \leftarrow X_{\beta t} \left[ \frac{\sum_\alpha W_{\alpha\beta} \left( Y_{\alpha t}/\tilde{Y}_{\alpha t} \right)}{\sum_\gamma W_{\gamma\beta}} \right]. \tag{6}$$

These alternating updates are guaranteed to converge to a local minimum of the total reconstruction error; see [11] for further details.

In our application of NMF to pitch estimation, the vectors $\mathbf{y}_t$ store vertical "time slices" of the IF representation in Fig. 1. Specifically, the elements of $\mathbf{y}_t$ store the magnitude spectra $|F(\omega_i^*(t), t)|$ of extracted partials at time $t$; the instantaneous frequency axis is discretized on a log scale so that each element of $\mathbf{y}_t$ covers 1/36 octave of the frequency spectrum. The columns of $\mathbf{W}$ store basis functions, or harmonic templates, for the magnitude spectra of voiced speech with different fundamental frequencies. (An additional column in $\mathbf{W}$ stores a non-harmonic template for unvoiced speech.) In this study, only one harmonic template was used per fundamental frequency. The fundamental frequencies range from 50Hz to 400Hz, spaced and discretized on a log scale. We constrained the harmonic templates for different fundamental frequencies to be related by a simple translation on the log-frequency axis. Tying the columns of $\mathbf{W}$ in this way greatly reduces the number of parameters that must be estimated by a learning algorithm. Finally, the elements of $\mathbf{x}_t$ store the coefficients that best reconstruct $\mathbf{y}_t$ by linearly superposing harmonic templates of $\mathbf{W}$. Note that only partials from the same source form harmonic relations. Thus, the number of nonzero elements in $\mathbf{x}_t$ indicates the number of periodic sources at time $t$, while the indices of nonzero elements indicate their fundamental frequencies. It is in this sense that the reconstruction $\mathbf{y}_t \approx \mathbf{W}\mathbf{x}_t$ provides an analysis of the auditory scene.

### 3.1 Learning the basis functions of voiced speech

The harmonic templates in $\mathbf{W}$ were estimated from the voiced speech of (non-overlapping) speakers in the Keele database [14]. The Keele database provides aligned pitch contours derived from laryngograph recordings. The first halves of all utterances were used for training, while the second halves were reserved for testing. Given the vectors $\mathbf{y}_t$ computed by IF estimation in the front end, the problem of NMF is to estimate the columns of $\mathbf{W}$ and the reconstruction coefficients $\mathbf{x}_t$. Each $\mathbf{x}_t$ has only two nonzero elements (one indicating the reference value for $f_0$, the other corresponding to the non-harmonic template of the basis matrix $\mathbf{W}$); their magnitudes must still be estimated by NMF. The estimation was performed by iterating the updates in eq. (6).

Fig. 2 (left) compares the harmonic template at 100 Hz before and after learning. While the template is initialized with broad spectral peaks, it is considerably sharpened by the NMF learning algorithm. Fig. 2 (right) shows four examples from the Keele database (from snippets of voiced speech with $f_0 = 100$ Hz) that were used to train this template. Note that even among these four partial profiles there is considerable variance. The learned template is derived to minimize the total reconstruction error over all segments of voiced speech in the training data.

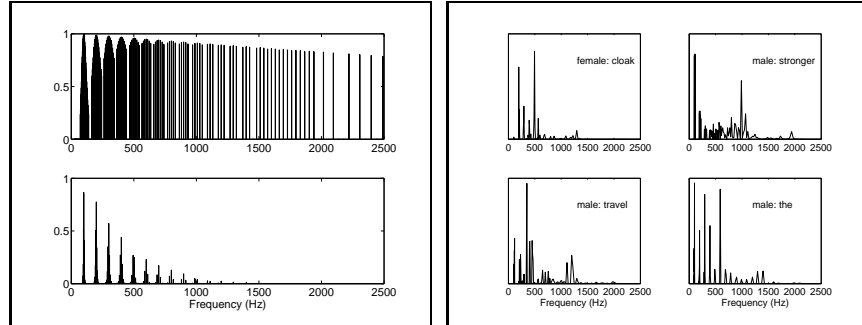

Figure 2: *Left:* harmonic template before and after learning for voiced speech at $f_0 = 100$ Hz. The learned template (bottom) has a much sharper spectral profile. *Right:* observed partials from four speakers with $f_0 = 100$ Hz.

### 3.2 Nonnegative deconvolution for estimating $f_0$ of one or more voices

Once the basis functions in $\mathbf{W}$ have been estimated, computing $\mathbf{x}$ such that $\mathbf{y} \approx \mathbf{Wx}$ under the measure of eq. (5) simplifies to the problem of nonnegative deconvolution. Nonnegative deconvolution has been applied to problems in fundamental frequency estimation [16], music analysis [17] and sound localization [12].

In our model, nonnegative deconvolution of $\mathbf{y} \approx \mathbf{Wx}$ yields an estimate of the number of periodic sources in $\mathbf{y}$ as well as their $f_0$ values. Ideally, the number of nonzero reconstruction weights in $\mathbf{x}$ reveal the number of sources, and the corresponding columns in the basis matrix $\mathbf{W}$ reveal their $f_0$ values. In practice, the index of the largest component of $\mathbf{x}$ is found, and its corresponding $f_0$ value is deemed to be the dominant fundamental frequency. The second largest component of $\mathbf{x}$ is then used to extract a secondary fundamental frequency, and so on. A thresholding heuristic can be used to terminate the search for additional sources. Unvoiced speech is detected by a simple frame-based classifier trained to make voiced/unvoiced distinctions from the observation $\mathbf{y}$ and its nonnegative deconvolution $\mathbf{x}$.

The pattern-matching computations in NMF are reminiscent of well-known models of harmonic template matching [6]. Two main differences are worth noting. First, the templates in NMF are learned from labeled speech data. We have found this to be essential in their generalization to unseen cases. It is not obvious how to craft a harmonic template "by hand" that manages the variability of partial profiles in Fig. 2 (right). Second, the template matching in NMF is framed by nonnegativity constraints. Specifically, the algorithm models observed partials by a *nonnegative* superposition of harmonic stacks. The cost function in eq. (5) also diverges if $\tilde{y}_\alpha = 0$ when $y_\alpha$ is nonzero; this useful property ensures that *minima of eq. (5) must explain each observed partial by its attribution to one or more sources.* This property does not hold for traditional least-squares linear reconstructions.

## 4 Implementation and results

We have implemented both the IF estimation in section 2 and the nonnegative deconvolution in section 3.2 in a real-time system for pitch tracking. The software runs on a laptop computer with a visual display that shows the contour of estimated $f_0$ values scrolling in real-time. After the signal is downsampled to 4900 Hz, IF estimation is performed in 10 ms shifts with an analysis window of 50 ms. Partials extracted from the fixed points of eq. (4) are discretized on a log-frequency axis. The columns of the basis matrix $\mathbf{W}$ provide har-

Keele database

|  | VE (%) | UE (%) | GPE (%) | RMS (Hz) |
|---|---|---|---|---|
| NMF | 7.7 | 4.6 | 0.9 | 4.3 |
| RAPT | 3.2 | 6.8 | 2.2 | 4.4 |

Edinburgh database

|  | VE (%) | UE (%) | GPE (%) | RMS (Hz) |
|---|---|---|---|---|
| NMF | 7.8 | 4.4 | 0.7 | 5.8 |
| RAPT | 4.5 | 8.4 | 1.9 | 5.3 |

Table 1: Comparison between our algorithm and RAPT [18] on the test portion of the Keele database (see text) and the full Edinburgh database, in terms of the percentages of voiced errors (VE), unvoiced errors (UE), and gross pitch errors (GPE), as well as the root mean square (RMS) deviation in Hz.

monic templates for $f_0 = 50$ Hz to $f_0 = 400$ Hz with a step size of $1/36$ octave. To achieve real-time performance and reduce system latency, the system does not postprocess the $f_0$ values obtained in each frame from nonnegative deconvolution: in particular, there is no dynamic programming to smooth the pitch contour, as commonly done in many pitch tracking algorithms [18]. We have found that our algorithm performs well and yields smooth pitch contours (for non-overlapping voices) even without this postprocessing.

## 4.1 Pitch determination of clean speech signals

Table 1 compares the performance of our algorithm on clean speech to RAPT [18], a state-of-the-art pitch tracker based on autocorrelation and dynamic programming. Four error types are reported: the percentage of voiced frames misclassified as unvoiced (VE), the percentage of unvoiced frames misclassified as voiced (UE), the percentage of voiced frames with gross pitch errors (GPE) where predicted and reference $f_0$ values differ by more than 20%, and the root-mean-squared (RMS) difference between predicted and reference $f_0$ values when there are no gross pitch errors. The results were obtained on the second halves of utterances reserved for testing in the Keele database, as well as the full set of utterances in the Edinburgh database [3]. As shown in the table, the performance of our algorithm is comparable to that of RAPT.

## 4.2 Pitch determination of overlapping voices and noisy speech

We have also examined the robustness of our system to noise and overlapping speakers. Fig. 3 shows the $f_0$ values estimated by our algorithm from a mixture of two voices—one with ascending pitch, the other with descending pitch. Each voice spans one octave. The dominant and secondary $f_0$ values extracted in each frame by nonnegative deconvolution are shown. The algorithm recovers the $f_0$ values of the individual voices almost perfectly, though it does not currently make any effort to track the voices through time. (This is a subject for future work.)

Fig. 4 shows in more detail how IF estimation and nonnegative deconvolution are affected by interfering speakers and noise. A clean signal from a single speaker is shown in the top row of the plot, along with its log power spectra, partials extracted by IF estimation, estimated $f_0$, and reconstructed harmonic stack. The second and third rows show the effects of adding white noise and an overlapping speaker, respectively. Both types of interference degrade the harmonic structure in the log power spectra and extracted partials. However, nonnegative deconvolution is still able to recover the pitch of the original speaker, as well as the pitch of the second speaker. On larger evaluations of the algorithm's robustness, we have obtained results comparable to RAPT over a wide range of SNRs (as low as 0 dB).

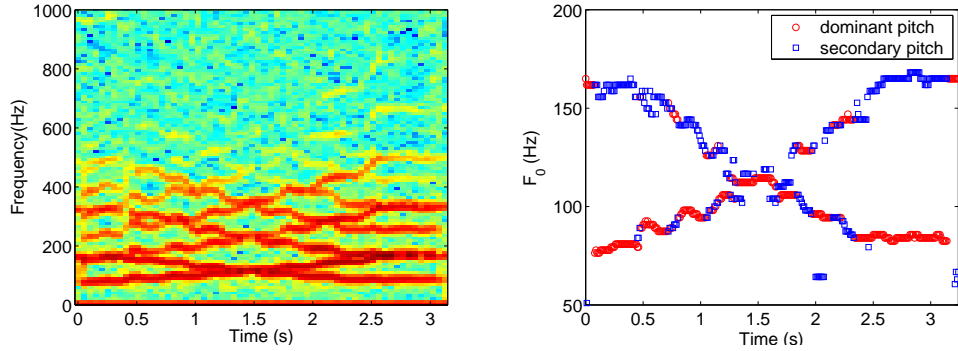

Figure 3: *Left:* Spectrogram of a mixture of two voices with ascending and descending $f_0$ contours. *Right:* $f_0$ values estimated by NMF.

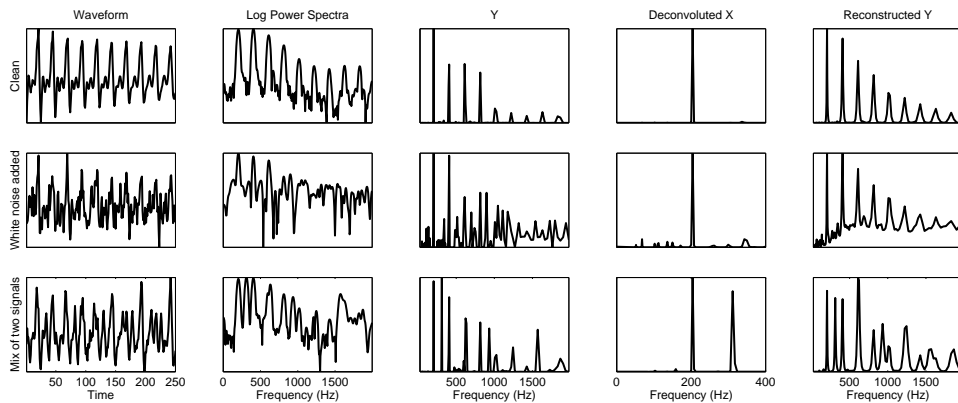

Figure 4: Effect of white noise (middle row) and overlapping speaker (bottom row) on clean speech (top row). Both types of interference degrade the harmonic structure in the log power spectra (second column) and the partials extracted by IF estimation (third column). The results of nonnegative deconvolution (fourth column), however, are fairly robust. Both the pitch of the original speaker at $f_0 = 200\,\text{Hz}$ and the overlapping speaker at $f_0 = 300\,\text{Hz}$ are recovered. The fifth column displays the reconstructed profile of extracted partials from activated harmonic templates.

## 5 Discussion

There exists a large body of related work on fundamental frequency estimation of overlapping sources [5, 7, 9, 19, 20, 21]. Our contributions in this paper are to develop a new framework based on recent advances in unsupervised learning and to study the problem with the constraints imposed by real-time system building. Nonnegative deconvolution is similar to EM algorithms [7] for harmonic template matching, but it does not impose normalization constraints on spectral peaks as if they represented a probability distribution. Important directions for future work are to train a richer set of harmonic templates by NMF, to incorporate the frame-based computations of nonnegative deconvolution into a dynamical model, and to embed our real-time system in interactive agents that respond to the pitch contours of human speech. All these directions are being actively pursued.

# References

[1] T. Abe, T. Kobayashi, and S. Imai. Harmonics tracking and pitch extraction based on instantaneous frequency. In *Proc. of ICASSP*, pages 756–759, 1995.

[2] T. Abe, T. Kobayashi, and S. Imai. Robust pitch estimation with harmonics enhancement in noisy environments based on instantaneous frequency. In *Proc. of ICSLP*, pages 1277–1280, 1996.

[3] P. Bagshaw, S. M. Hiller, and M. A. Jack. Enhanced pitch tracking and the processing of f0 contours for computer aided intonation teaching. In *Proc. of 3rd European Conference on Speech Communication and Technology*, pages 1003–1006, 1993.

[4] A. S. Bregman. *Auditory Scene Analysis: The Perceptual Organization of Sound*. MIT Press, 2nd edition, 1999.

[5] A. de Cheveigne and H. Kawahara. Multiple period estimation and pitch perception model. *Speech Communication*, 27:175–185, 1999.

[6] J. Goldstein. An optimum processor theory for the central formation of the pitch of complex tones. *J. Acoust. Soc. Am.*, 54:1496–1516, 1973.

[7] M. Goto. A robust predominant-F0 estimation method for real-time detection of melody and bass lines in CD recordings. In *Proc. of ICASSP*, pages 757–760, June 2000.

[8] H. Kawahara, H. Katayose, A. de Cheveigné, and R. D. Patterson. Fixed point analysis of frequency to instantaneous frequency mapping for accurate estimation of f0 and periodicity. In *Proc. of EuroSpeech*, pages 2781–2784, 1999.

[9] A. Klapuri, T. Virtanen, and J.-M. Holm. Robust multipitch estimation for the analysis and manipulation of polyphonic musical signals. In *Proc. of COST-G6 Conference on Digital Audio Effects*, Verona, Italy, 2000.

[10] D. D. Lee and H. S. Seung. Learning in intelligent embedded systems. In *Proc. of USENIX Workshop on Embedded Systems*, 1999.

[11] D. D. Lee and H. S. Seung. Learning the parts of objects with nonnegative matrix factorization. *Nature*, 401:788–791, 1999.

[12] Y. Lin, D. D. Lee, and L. K. Saul. Nonnegative deconvolution for time of arrival estimation. In *Proc. of ICASSP*, 2004.

[13] T. Nakatani and T. Irino. Robust fundamental frequency estimation against background noise and spectral distortion. In *Proc. of ICSLP*, pages 1733–1736, 2002.

[14] F. Plante, G. F. Meyer, and W. A. Ainsworth. A pitch extraction reference database. In *Proc. of EuroSpeech*, pages 837–840, 1995.

[15] L. K. Saul, D. D. Lee, C. L. Isbell, and Y. LeCun. Real time voice processing with audiovisual feedback: toward autonomous agents with perfect pitch. In S. Becker, S. Thrun, and K. Obermayer, editors, *Advances in Neural Information Processing Systems 15*. MIT Press, 2003.

[16] L. K. Saul, F. Sha, and D. D. Lee. Statistical signal processing with nonnegativity constraints. In *Proc. of EuroSpeech*, pages 1001–1004, 2003.

[17] P. Smaragdis and J. C. Brown. Non-negative matrix factorization for polyphonic music transcription. In *Proc. of IEEE Workshop on Applications of Signal Processing to Audio and Acoustics*, pages 177–180, 2003.

[18] D. Talkin. A robust algorithm for pitch tracking(RAPT). In W. B. Kleijn and K. K. Paliwal, editors, *Speech Coding and Synthesis*, chapter 14. Elsevier Science B.V., 1995.

[19] T. Tolonen and M. Karjalainen. A computationally efficient multipitch analysis model. *IEEE Trans. on Speech and Audio Processing*, 8(6):708–716, 2000.

[20] T. Virtanen and A. Klapuri. Separation of harmonic sounds using multipitch analysis and iterative parameter estimation. In *Proc. of IEEE Workshop on Applications of Signal Processing to Audio and Acoustics*, pages 83–86, New Paltz, NY, USA, Oct 2001.

[21] M. Wu, D. Wang, and G. J. Brown. A multipitch tracking algorithm for noisy speech. *IEEE Trans. on Speech and Audio Processing*, 11:229–241, 2003.
